# Class-size Independent Generalization Analsysis of Some Discriminative Multi-Category Classification Methods

**Tong Zhang**
IBM T.J. Watson Research Center
Yorktown Heights, NY 10598
tzhang@watson.ibm.com

## Abstract

We consider the problem of deriving class-size independent generalization bounds for some regularized discriminative multi-category classification methods. In particular, we obtain an expected generalization bound for a standard formulation of multi-category support vector machines. Based on the theoretical result, we argue that the formulation over-penalizes misclassification error, which in theory may lead to poor generalization performance. A remedy, based on a generalization of multi-category logistic regression (conditional maximum entropy), is then proposed, and its theoretical properties are examined.

## 1 Introduction

We consider the multi-category classification problem, where we want to find a predictor $p : \mathcal{X} \to \mathcal{Y}$, where $\mathcal{X}$ is a set of possible inputs and $\mathcal{Y}$ is a discrete set of possible outputs. In many applications, the output space $\mathcal{Y}$ can be extremely large, and may be regarded as infinity for practical purposes. For example, in natural language processing and sequence analysis, the input can be an English sentence, and the output can be a parse or a translation of the sentence. For such applications, the number of potential outputs can be exponential of the length of the input sentence. As another example, in machine learning based web-page search and ranking, the input is the keywords and the output space consists of all web-pages.

In order to handle such application tasks, from the theoretical point of view, we do not need to assume that the output space $\mathcal{Y}$ is finite, so that it is crucial to obtain generalization bounds that are independent of the size of $\mathcal{Y}$. For such large scale applications, one often has a routine that maps each $x \in \mathcal{X}$ to a subset of candidates $\mathbf{GEN}(x) \subset \mathcal{Y}$, so that the desired output associated with $x$ belongs to $\mathbf{GEN}(x)$. For example, for web-page search, $\mathbf{GEN}(x)$ consists of all pages that contain one or more keywords in $x$. For sequence annotation, $\mathbf{GEN}(x)$ may include all annotation sequences that are consistent. Although the set $\mathbf{GEN}(x)$ may significantly reduce the size of potential outputs $\mathcal{Y}$, it can still be large. Therefore it is important that our learning bounds are independent of the size of $\mathbf{GEN}(x)$.

We consider the general setting of learning in Hilbert spaces since it includes the popular

kernel methods. Let our feature space $H$ be a reproducing kernel Hilbert space with dot product $\cdot$. For a weight vector $\mathbf{w} \in H$, we use notation $\|\mathbf{w}\|_H^2 = \mathbf{w} \cdot \mathbf{w}$. We associate each possible input/output pair $(x, y) \in \mathcal{X} \times \mathcal{Y}$ with a feature vector $\mathbf{f}_{x,y} \in H$. Our classifier is characterized by a weight vector $\mathbf{w} \in H$, with the following classification rule:

$$p_{\mathbf{w}}(x) = \arg \max_{c \in \mathbf{GEN}(x)} \mathbf{w} \cdot \mathbf{f}_{x,c}. \tag{1}$$

Note that computational issues are ignored in this paper. In particular, we assume that the above decision can be computed efficiently (either approximately or exactly) even when $\mathbf{GEN}(x)$ is large. In practice, this is often possible either by heuristic search or dynamic programming (when $\mathbf{GEN}(X)$ has certain local-dependency structures). In this paper, we are only interested in the learning performance, so that we will not discuss the computational aspect.

We assume that the input/output pair $(x, y) \in \mathcal{X} \times \mathcal{Y}$ is drawn from an unknown underlying distribution $D$. The quality of the predictor $\mathbf{w}$ is measured by some loss function. In this paper, we focus on the expected classification error with respect to $D$:

$$\ell_D(\mathbf{w}) = \mathbf{E}_{(X,Y)} \, I(p_{\mathbf{w}}(X), Y), \tag{2}$$

where $(X, Y)$ is drawn from $D$, and $I$ is the standard 0-1 classification error: $I(Y', Y) = 0$ when $Y' = Y$ and $I(Y', Y) = 1$ when $Y' \neq Y$.

The general set up we described above is useful for many application problems, and has been investigated, for example, in [2, 6]. The important issue of class-size independent (or weakly dependent) generalization analysis has also been discussed there.

Consider a set of training data $S = \{(X_i, Y_i), i = 1, \ldots, n\}$, where we assume that for each $i$, $Y_i \in \mathbf{GEN}(X_i)$. We would like to find $\hat{\mathbf{w}}_S \in H$ such that the classification error $\ell_D(\hat{\mathbf{w}}_S)$ is as small as possible. This paper studies regularized discriminative learning methods that estimate a weight vector $\hat{\mathbf{w}}_S \in H$ by solving the following optimization problem:

$$\hat{\mathbf{w}}_S = \arg \min_{w \in H} \left[ \frac{1}{n} \sum_{i=1}^{n} L(\mathbf{w}, X_i, Y_i) + \frac{\lambda}{2} \|\mathbf{w}\|_H^2 \right], \tag{3}$$

where $\lambda \geq 0$ is an appropriately chosen regularization parameter, and $L(\mathbf{w}, X, Y)$ is a loss function which is convex of $\mathbf{w}$. In this paper, we focus on some loss functions of the following form:

$$L(\mathbf{w}, X, Y) = \psi \left( \sum_{c \in \mathbf{GEN}(X) \backslash Y} \phi(\mathbf{w} \cdot (\mathbf{f}_{X,Y} - \mathbf{f}_{X,c})) \right),$$

where $\psi$ and $\phi$ are appropriately chosen real-valued functions.

Typically $\psi$ is chosen as an increasing function and $\phi$ as a decreasing function, selected so that (3) is a convex optimization problem. The intuition behind this method is that the resulting optimization formulation favors large values $\mathbf{w} \cdot (\mathbf{f}_{X_i,Y_i} - \mathbf{f}_{X_i,c})$ for all $c \in \mathbf{GEN}(X_i) \backslash Y_i$. Therefore, it favors a weight vector $\mathbf{w} \in H$ such that $\mathbf{w} \cdot \mathbf{f}_{X_i,Y_i} = \arg \max_{c \in \mathbf{GEN}(X_i)} \mathbf{w} \cdot \mathbf{f}_{X_i,c}$, which encourages the correct classification rule in (1). The regularization term $\frac{\lambda}{2} \|\mathbf{w}\|_H^2$ is included for capacity control, which has become the standard practice in machine learning nowadays.

Two of the most important methods used in practice, multi-category support vector machines [7] and penalized multi-category logistic regression (conditional maximum entropy with Gaussian smoothing [1]), can be regarded as special cases of (3). The purpose of this paper is to study their generalization behaviors. In particular, we are interested in generalization bounds that are independent of the size of $\mathbf{GEN}(X_i)$.

## 2 Multi-category Support Vector Machines

We consider the multi-category support vector machine method proposed in [7]. It is a special case of (3) with $\hat{\mathbf{w}}_S$ computed based on the following formula:

$$\hat{\mathbf{w}}_S = \arg\min_{w \in H} \left[ \frac{1}{n} \sum_{i=1}^{n} \sum_{c \in \mathbf{GEN}(X_i) \backslash Y_i} h(\mathbf{w} \cdot (\mathbf{f}_{X_i,Y_i} - \cdot \mathbf{f}_{X_i,c})) + \frac{\lambda}{2} \|\mathbf{w}\|_H^2 \right], \quad (4)$$

where $h(z) = \max(1 - z, 0)$ is the hinge loss used in the standard SVM formulation. From the asymptotic statistical point of view, this formulation has some drawbacks in that there are cases such that the method does not lead to a classifier that achieves the Bayes error [9] (inconsistency). A Bayes consistent remedy has been proposed in [4]. However, method based on (4) has some attractive properties, and has been successfully used for some practical problems.

We are interested in the generalization performance of (4). As we shall see, this formulation performs very well in the linearly separable (or near separable) case. Our analysis also reveals a problem of this method for non-separable problems. Specifically, the formulation over-penalizes classification error. Possible remedies will be suggested at the end of the section.

We start with the following theorem, which specifies a generalization bound in a form often referred to as the *oracle inequality*. That is, it bounds the generalization performance of the SVM method (4) in terms of the best possible true multi-category SVM loss. Proof is left to Appendix B.

**Theorem 2.1** *Let $M = \sup_X \sup_{Y,Y' \in \mathbf{GEN}(X)} \|\mathbf{f}_{X,Y} - \mathbf{f}_{X,Y'}\|_H$. The expected generalization error of (4) can be bounded as:*

$$\mathbf{E}_S \ell_D(\hat{\mathbf{w}}_S) \leq \mathbf{E}_S \mathbf{E}_{(X,Y)} \sup_{c \in \mathbf{GEN}(X) \backslash Y} h(\hat{\mathbf{w}}_S \cdot (\mathbf{f}_{X,Y} - \mathbf{f}_{X,c}))$$

$$\leq \frac{\max(\lambda n, M^2) + M^2}{\lambda n} \inf_{w \in H} \left[ \mathbf{E}_{(X,Y)} \sum_{c \in \mathbf{GEN}(X) \backslash Y} h(\mathbf{w} \cdot (\mathbf{f}_{X,Y} - \mathbf{f}_{X,c})) + \frac{\lambda n \|\mathbf{w}\|_H^2}{2(n+1)} \right],$$

*where $\mathbf{E}_S$ is the expectation with respect to the training data.*

Note that the generalization bound does not depend on the size of $\mathbf{GEN}(X)$, which is what we want to achieve. The left-hand side of the theorem bounds the classification error of the multi-category SVM classifier in terms of $\sup_{c \in \mathbf{GEN}(X) \backslash Y} h(\hat{\mathbf{w}}_S \cdot (\mathbf{f}_{X,Y} - \mathbf{f}_{X,c}))$, while the right hand side in terms of $\sum_{c \in \mathbf{GEN}(X) \backslash Y} h(\mathbf{w} \cdot (\mathbf{f}_{X,Y} - \mathbf{f}_{X,c}))$. There is a mismatch here. The latter is a very loose bound since it over-counts classification errors in the summation when multiple errors are made at the same point. In fact, although the class-size dependency does not come into our generalization analysis, it may well come into the summation term $\sum_{c \in \mathbf{GEN}(X) \backslash Y} h(\mathbf{w} \cdot (\mathbf{f}_{X,Y} - \mathbf{f}_{X,c}))$ when multiple errors are made at the same point. We believe that this is a serious flaw of the method, which we will try to remedy later. However, the bound can be quite tight in the near separable case, when $\sum_{c \in \mathbf{GEN}(X) \backslash Y} h(\hat{\mathbf{w}}_S \cdot (\mathbf{f}_{X,Y} - \mathbf{f}_{X,c}))$ is small. The following Corollary gives such a result:

**Corollary 2.1** *Assume that there is a large margin separator $\mathbf{w}_* \in H$ such that for each data point $(X, Y)$, the following margin condition holds:*

$$\forall c \in \mathbf{GEN}(X) \backslash Y : w_* \cdot \mathbf{f}_{X,Y} \geq w_* \cdot \mathbf{f}_{X,c} + 1.$$

*Then in the limit of $\lambda \to 0$, the expected generalization error of (4) can be bounded as:*

$$\mathbf{E}_S \ell_D(\hat{\mathbf{w}}_S) \leq \frac{\|w_*\|_H^2}{n+1} \sup_X \sup_{Y,Y' \in \mathbf{GEN}(X)} \|\mathbf{f}_{X,Y} - \mathbf{f}_{X,Y'}\|_H^2,$$

*where $\mathbf{E}_S$ is the expectation with respect to the training data.*

*Proof.* Just choose $\mathbf{w}_*$ on the right hand side of Theorem 2.1. $\square$

The above result for (4) gives a class-size independent bound for large margin separable problems. The bound generalizes a similar result for two-class hard-margin SVM. It also matches a bound for multi-class perceptron in [2]. To our knowledge, this is the first result showing that the generalization performance of a batch large margin algorithm such as (4) can be class-size independent (at least in the separable case). Previous results in [2, 6], relying on the covering number analysis, lead to bounds that depend on the size of $\mathcal{Y}$ (although the result in [6] is of a different style).

Our analysis also implies that the multi-category classification method (4) has good generalization behavior for separable problems. However, as pointed out earlier, for non-separable problems, the formulation over-penalize classification error since in the summation, it may count classification error at a point multiple times when multiple mistakes are made at the point. A remedy is to replace the summation symbol $\sum_{c \in \mathbf{GEN}(X_i) \setminus Y_i}$ in (4) by the sup operator $\sup_{c \in \mathbf{GEN}(X_i) \setminus Y_i}$, as we have used for bounding the classification error on the left hand side of Theorem 2.1. This is done in [3]. However, like (4), the resulting formulation is also inconsistent. Instead of using a hard-sup operator, we may also use a soft-sup operator, which can possibly lead to consistency. For example, consider the equality $\sup_c |h_c| = \lim_{p \to \infty} (\sum_c |h_c|^p)^{1/p}$, we may approximate the right hand side limit with a large $p$. Another more interesting formulation is to consider $\sup_c h_c = \lim_{p \to \infty} p^{-1} \ln(\sum_c \exp(ph_c))$, which leads to a generalization of the conditional maximum entropy method.

## 3   Large Margin Discriminative Maximum Entropy Method

Based on the motivation given at the end of the last section, we propose the following generalization of maximum entropy (multi-category logistic regression) with Gaussian prior (see [1]). It introduces a margin parameter into the standard maximum entropy formulation, and can be regarded as a special case of (3):

$$\hat{\mathbf{w}}_S = \arg \min_{w \in H} \left[ \frac{1}{n} \sum_{i=1}^n \frac{1}{p} \ln \left( 1 + \sum_{c \in \mathbf{GEN}(X_i) \setminus Y_i} e^{p(\gamma - \mathbf{w} \cdot (\mathbf{f}_{X_i,Y_i} - \mathbf{f}_{X_i,c}))} \right) + \frac{\lambda}{2} \|\mathbf{w}\|_H^2 \right], \quad (5)$$

where $\gamma$ is a margin condition, and $p > 0$ is a scaling factor (which in theory can also be removed by a redefinition of $\mathbf{w}$ and $\gamma$).

If we choose $\gamma = 0$, then this formulation is equivalent to the standard maximum entropy method. If we pick the margin parameter $\gamma = 1$, and let $p \to \infty$, then

$$\frac{1}{p} \ln \left( 1 + \sum_{c \in \mathbf{GEN}(X_i) \setminus Y_i} e^{p(\gamma - \mathbf{w} \cdot (\mathbf{f}_{X_i,Y_i} - \mathbf{f}_{X_i,c}))} \right) \to \sup_{c \in \mathbf{GEN}(X_i) \setminus Y_i} h(\mathbf{w} \cdot (\mathbf{f}_{X_i,Y_i} - \mathbf{f}_{X_i,c})),$$

where $h(z) = \max(0, 1 - z)$ is used in (4). In this case, the formulation reduces to (4) but with $\sum_{c \in \mathbf{GEN}(X_i) \setminus Y_i}$ replaced by $\sup_{c \in \mathbf{GEN}(X_i) \setminus Y_i}$. As discussed at the end of last section, this solves the problem of over-counting the classification error.

In general, even with a finite scaling factor $p$, the log-transform in (4) guarantees that one penalizes misclassification error at most $\frac{1}{p} \ln |\mathbf{GEN}(X_i)|$ times at a point, where

$|\mathbf{GEN}(X_i)|$ is the size of $\mathbf{GEN}(X_i)$, while in (4), one may potentially over-penalize $|\mathbf{GEN}(X_i)|$ times. Clearly this is a desirable effect for non-separable problems. Methods in (5) have many attractive properties. In particular, we are able to derive class-size independent generalization bounds for this method. The proof of the following theorem is given in Appendix C.

**Theorem 3.1** *Let $M = \sup_X \sup_{Y,Y' \in \mathbf{GEN}(X)} \|\mathbf{f}_{X,Y} - \mathbf{f}_{X,Y'}\|_H$. Define loss $L(\mathbf{w}, x, y)$ as:*

$$L(\mathbf{w}, x, y) = \frac{1}{p} \ln \left( 1 + \sum_{c \in \mathbf{GEN}(x) \backslash y} e^{p(\gamma - \mathbf{w} \cdot (\mathbf{f}_{x,y} - \mathbf{f}_{x,c}))} \right),$$

*and let*

$$Q_\lambda = \inf_{w \in H} \left[ \mathbf{E}_{(X,Y)} L(\mathbf{w}, X, Y) + \frac{\lambda n}{2(n+1)} \|\mathbf{w}\|_H^2 \right].$$

*The expected generalization error of (5) can be bounded as:*

$$\mathbf{E}_S \mathbf{E}_{(X,Y)} L(\hat{\mathbf{w}}_S, X, Y) \leq Q_\lambda + \frac{M^2}{\lambda n} (1 - e^{-pQ_\lambda}).$$

*where $\mathbf{E}_S$ is the expectation with respect to the training data.*

Theorem 3.1 gives a class-size independent generalization bound for (5). Note that the left hand side is the true loss of the $\hat{\mathbf{w}}_S$ from (5), and the right hand size is specified in terms of the best possible regularized true loss $Q_\lambda$, plus a penalty term that is no larger than $M^2/(\lambda n)$. It is clear that this generalization bound is class-size independent. Moreover, unlike Theorem 2.1, the loss function on the left hand side matches the loss function on the right hand side in Theorem 3.1. These are not trivial properties. In fact, most learning methods do not have these desirable properties. We believe this is a great advantage for the maximum entropy-type discriminative learning method in (5). It implies that this class of algorithms are suitable for problems with large number of classes. Moreover, we can see that the generalization performance is well-behaved no matter what values of $p$ and $\gamma$ we choose.

If we take $\gamma = 0$ and $p = 1$, then we obtain a generalization bound for the popular maximum entropy method with Gaussian prior, which has been widely used in natural language processing applications. To our knowledge, this is the first generalization bound derived for this method. Our result not only shows the importance of Gaussian prior regularization, but also implies that the regularized conditional maximum entropy method has very desirable generalization behavior.

Another interesting special case of (5) is to let $\gamma = 1$ and $p \to \infty$. For simplicity we only consider the case that $|\mathbf{GEN}(X)|$ is finite (but can be arbitrarily large). In this case, we note that $0 \leq L(\mathbf{w}, X, Y) - \sup_{c \in \mathbf{GEN}(X) \backslash Y} h(\mathbf{w} \cdot (\mathbf{f}_{X,Y} - \mathbf{f}_{X,c})) \leq \frac{\ln |\mathbf{GEN}(X)|}{p}$. We thus obtain from Theorem 3.1 a bound

$$\mathbf{E}_S \mathbf{E}_{(X,Y)} \sup_{c \in \mathbf{GEN}(X) \backslash Y} h(\hat{\mathbf{w}}_S \cdot (\mathbf{f}_{X,Y} - \mathbf{f}_{X,c})) \leq \frac{\mathbf{E}_X \ln |\mathbf{GEN}(X)|}{p} + \frac{M^2}{\lambda n}$$

$$+ \inf_{w \in H} \left[ \mathbf{E}_{(X,Y)} \sup_{c \in \mathbf{GEN}(X) \backslash Y} h(\mathbf{w} \cdot (\mathbf{f}_{X,Y} - \mathbf{f}_{X,c})) + \frac{\lambda \|\mathbf{w}\|_H^2}{2} \right].$$

Now we can take a sufficiently large $p$ such that the term $\mathbf{E}_X \ln |\mathbf{GEN}(X)|/p$ becomes negligible. Let $p \to \infty$, the result implies a bound for the SVM method in [3]. For non-separable problems, this bound is clearly superior to the SVM bound in Theorem 2.1 since the right hand side replaces the summation $\sum_{c \in \mathbf{GEN}(X) \backslash Y}$ by the sup operator

$\sup_{c \in \mathbf{GEN}(X) \setminus Y}$. In theory, this satisfactorily solves the problem of over-penalizing mis-classification error. Moreover, an advantage over [3] is that for some $p$, consistency can be achieved. Our analysis also establishes a bridge between the Gaussian smoothed maximum entropy method [1] and the SVM method in [3].

## 4   Conclusion

We studied the generalization performance of some regularized multi-category classification methods. In particular, we derived a class-size independent generalization bound for a standard formulation of multi-category support vector machines. Based on the theoretical investigation, we showed that this method works well for linearly separable problems. However, it over-penalizes mis-classification error, leading to loose generalization bounds in the non-separable case. A remedy, based on a generalization of the maximum entropy method, is proposed. Moreover, we are able to derive class-size independent bounds for the newly proposed formulation, which implies that this class of methods (including the standard maximum entropy) are suitable for classification problems with very large number of classes. We showed that in theory, the new formulation provides a satisfactory solution to the problem of over-penalizing mis-classification error.

## A   A general stability bound

The following lemma is essentially a variant of similar stability results for regularized learning systems used in [8, 10]. We include the proof Sketch for completeness.

**Lemma A.1** *Consider a sequence of convex functions $L_i(\mathbf{w})$ for $i = 1, 2, \ldots$ Define for $k = 1, 2, \ldots$*

$$\mathbf{w}_k = \arg\min_{\mathbf{w}} \left[ \sum_{i=1}^{k} L_i(\mathbf{w}) + \frac{\lambda n}{2} \|\mathbf{w}\|_H^2 \right].$$

*Then for all $k \geq 1$, there exists subgradient (cf. [5]) $\nabla L_{k+1}(\mathbf{w}_{k+1})$ of $L_i$ at $\mathbf{w}_{k+1}$ such that*

$$\mathbf{w}_{k+1} = -\frac{1}{\lambda n} \sum_{i=1}^{k+1} \nabla L_i(\mathbf{w}_{k+1}), \quad \|\mathbf{w}_k - \mathbf{w}_{k+1}\|_H \leq \frac{1}{\lambda n} \|\nabla L_{k+1}(\mathbf{w}_{k+1})\|_H.$$

*Proof Sketch.* The first equality is the first-order condition for the optimization problem [5] where $\mathbf{w}_{k+1}$ is the solution. Now, subtracting this equality at $\mathbf{w}_k$ and $\mathbf{w}_{k+1}$, we have:

$$-\lambda n(\mathbf{w}_{k+1} - \mathbf{w}_k) = \nabla L_{k+1}(\mathbf{w}_{k+1}) + \sum_{i=1}^{k} (\nabla L_i(\mathbf{w}_{k+1}) - \nabla L_i(\mathbf{w}_k)).$$

Multiply the two sides by $\mathbf{w}_{k+1} - \mathbf{w}_k$, we obtain

$$-\lambda n \|\mathbf{w}_{k+1} - \mathbf{w}_k\|_H^2 = \nabla L_{k+1}(\mathbf{w}_{k+1}) \cdot (\mathbf{w}_{k+1} - \mathbf{w}_k) + \sum_{i=1}^{k} (\nabla L_i(\mathbf{w}_{k+1}) - \nabla L_i(\mathbf{w}_k)) \cdot (\mathbf{w}_{k+1} - \mathbf{w}_k).$$

Note that $\nabla L_i(\mathbf{w}_{k+1}) - \nabla L_i(\mathbf{w}_k)) \cdot (\mathbf{w}_{k+1} - \mathbf{w}_k) = d_{L_i}(\mathbf{w}_k, \mathbf{w}_{k+1}) + d_{L_i}(\mathbf{w}_{k+1}, \mathbf{w}_k)$, where $d_L(\mathbf{w}, \mathbf{w}') = L(\mathbf{w}') - L(\mathbf{w}) - \nabla L(\mathbf{w}) \cdot (\mathbf{w}' - \mathbf{w})$ is often called the Bregman divergence of $L$, which is well-known to be non-negative for any convex function $L$ (this claim is also easy to verify by definition). We thus have $(\nabla L_i(\mathbf{w}_{k+1}) - \nabla L_i(\mathbf{w}_k)) \cdot (\mathbf{w}_{k+1} - \mathbf{w}_k) \geq 0$. It follows that

$$-\lambda n \|\mathbf{w}_{k+1} - \mathbf{w}_k\|_H^2 \geq \nabla L_{k+1}(\mathbf{w}_{k+1}) \cdot (\mathbf{w}_{k+1} - \mathbf{w}_k) \geq -\|\nabla L_{k+1}(\mathbf{w}_{k+1})\|_H \|\mathbf{w}_{k+1} - \mathbf{w}_k\|_H.$$

By canceling the factor $\|\mathbf{w}_{k+1} - \mathbf{w}_k\|_H$, we obtain the second inequality. $\square$

# B Proof Sketch of Theorem 2.1

Consider training samples $(X_i, Y_i)$ for $i = 1, \ldots, n+1$. Let $\tilde{\mathbf{w}}^k$ be the solution of (4) with the training sample $(X_k, Y_k)$ removed from the set (that is, the summation is $\sum_{i=1, i \neq k}^{n+1}$), and let $\tilde{\mathbf{w}}$ be the solution of (4) but with the summation $\sum_{i=1}^{n}$ replaced by $\sum_{i=1}^{n+1}$. Now for notation simplicity, we let $z_{k,c} = \tilde{\mathbf{w}} \cdot (\mathbf{f}_{X_k, Y_k} - \mathbf{f}_{X_k, c})$ for $c \in \mathbf{GEN}(X)$. It follows from Lemma A.1 that

$$\|\tilde{\mathbf{w}}\|_H^2 = -\frac{1}{\lambda n} \sum_{k=1}^{n+1} \sum_{c \in \mathbf{GEN}(X)} h'(z_{k,c}) z_{k,c}, \quad \|\tilde{\mathbf{w}}^k - \tilde{\mathbf{w}}\|_H \leq -\frac{M}{\lambda n} \sum_{c \in \mathbf{GEN}(X) \setminus Y} h'(z_{k,c}),$$

where $h'(\cdot)$ denotes a subgradient of $h(\cdot)$. Therefore using the inequality $-h'(z) \leq h(z) - h'(z)z$, we have

$$\sup_{c \in \mathbf{GEN}(X_k) \setminus Y_k} [h(\tilde{\mathbf{w}}^k \cdot (\mathbf{f}_{X_k, Y_k} - \mathbf{f}_{X_k, c})) - h(z_{k,c})] \leq \|\tilde{\mathbf{w}}^k - \tilde{\mathbf{w}}\|_H M$$

$$\leq -\frac{M^2}{\lambda n} \sum_{c \in \mathbf{GEN}(X_k) \setminus Y_k} h'(z_{k,c}) \leq \frac{M^2}{\lambda n} \sum_{c \in \mathbf{GEN}(X_k) \setminus Y_k} [h(z_{k,c}) - h'(z_{k,c}) z_{k,c}].$$

Summing over $k = 1, \ldots, n+1$, we obtain

$$\sum_{k=1}^{n+1} \sup_{c \in \mathbf{GEN}(X_k) \setminus Y_k} [h(\tilde{\mathbf{w}}^k \cdot (\mathbf{f}_{X_k, Y_k} - \mathbf{f}_{X_k, c})) - h(z_{k,c})]$$

$$\leq \frac{M^2}{\lambda n} \sum_{c \in \mathbf{GEN}(X_k) \setminus Y_k} \sum_{k=1}^{n+1} [h(z_{k,c}) - h'(z_{k,c}) z_{k,c}]$$

$$= \frac{M^2}{\lambda n} \sum_{c \in \mathbf{GEN}(X_k) \setminus Y_k} \sum_{k=1}^{n+1} h(z_{k,c}) + \|\tilde{\mathbf{w}}\|_H^2 M^2.$$

Therefore given an arbitrary $\mathbf{w} \in H$, we have

$$\sum_{k=1}^{n+1} \sup_{c \in \mathbf{GEN}(X_k) \setminus Y_k} h(\tilde{\mathbf{w}}^k \cdot (\mathbf{f}_{X_k, Y_k} - \mathbf{f}_{X_k, c}))$$

$$\leq (1 + \frac{M^2}{\lambda n}) \sum_{c \in \mathbf{GEN}(X_k) \setminus Y_k} \sum_{k=1}^{n+1} h(z_{k,c}) + \|\tilde{\mathbf{w}}\|_H^2 M^2$$

$$\leq \max(1 + \frac{M^2}{\lambda n}, \frac{2M^2}{\lambda n}) \left[ \sum_{c \in \mathbf{GEN}(X_k) \setminus Y_k} \sum_{k=1}^{n+1} h(z_{k,c}) + \frac{\lambda n}{2} \|\tilde{\mathbf{w}}\|_H^2 \right]$$

$$\leq \max(1 + \frac{M^2}{\lambda n}, \frac{2M^2}{\lambda n}) \left[ \sum_{c \in \mathbf{GEN}(X_k) \setminus Y_k} \sum_{k=1}^{n+1} h(\mathbf{w} \cdot (\mathbf{f}_{X_k, Y_k} - \mathbf{f}_{X_k, c})) + \frac{\lambda n}{2} \|\mathbf{w}\|_H^2 \right].$$

Now, taking expectation with respect to the training data, we obtain the bound.

# C Proof Sketch of Theorem 3.1

Similar to the proof of Theorem 2.1, we consider training samples $(X_i, Y_i)$ for $i = 1, \ldots, n+1$. Let $\tilde{\mathbf{w}}^k$ be the solution of (5) with the training sample $(X_k, Y_k)$ removed

from the set (that is, the summation is $\sum_{i=1, i \neq k}^{n+1}$), and let $\tilde{\mathbf{w}}$ be the solution of (5) but with the summation $\sum_{i=1}^{n}$ replaced by $\sum_{i=1}^{n+1}$. It follows from Lemma A.1 that

$$\|\tilde{\mathbf{w}}^k - \tilde{\mathbf{w}}\|_H \leq \frac{1}{\lambda n} \|\nabla L(\tilde{\mathbf{w}}, X_k, Y_k)\|_H \leq \frac{M}{\lambda n}(1 - e^{-pL(\tilde{\mathbf{w}}, X_k, Y_k)}).$$

Therefore

$$L(\tilde{\mathbf{w}}^k, X_k, Y_k) - L(\tilde{\mathbf{w}}, X_k, Y_k) \leq \frac{M^2}{\lambda n}(1 - e^{-pL(\tilde{\mathbf{w}}, X_k, Y_k)}).$$

Now summing over $k$, we obtain

$$\frac{1}{n+1} \sum_{k=1}^{n+1} L(\tilde{\mathbf{w}}^k, X_k, Y_k) \leq \frac{1}{n+1} \sum_{k=1}^{n+1} L(\tilde{\mathbf{w}}, X_k, Y_k) + \frac{M^2}{\lambda n} \left(1 - \frac{1}{n+1} \sum_{k=1}^{n+1} e^{-pL(\tilde{\mathbf{w}}, X_k, Y_k)}\right).$$

Taking expectation with respect to the training data, and using the following Jensen's inequality:

$$-\mathbf{E}_S \frac{1}{n+1} \sum_{k=1}^{n+1} e^{-pL(\tilde{\mathbf{w}}, X_k, Y_k)} \leq -e^{-p\mathbf{E}_S \frac{1}{n+1} \sum_{k=1}^{n+1} L(\tilde{\mathbf{w}}, X_k, Y_k)},$$

we obtain

$$\mathbf{E}_S \mathbf{E}_{(X_k, Y_k)} L(\tilde{\mathbf{w}}^k, X_k, Y_k) \leq \mathbf{E}_S \sum_{k=1}^{n+1} \frac{L(\tilde{\mathbf{w}}, X_k, Y_k)}{n+1} + \frac{M^2}{\lambda n} \left(1 - e^{-p\mathbf{E}_S \sum_{k=1}^{n+1} \frac{L(\tilde{\mathbf{w}}, X_k, Y_k)}{n+1}}\right).$$

Now, using the fact $\mathbf{E}_S \sum_{k=1}^{n+1} L(\tilde{\mathbf{w}}, X_k, Y_k) \leq (n+1)Q_\lambda$ (which follows from the optimal property of $\tilde{\mathbf{w}}$), we obtain the theorem.

# References

[1] Stanley Chen and Ronald Rosenfeld. A survey of smoothing techniques for ME models. *IEEE Trans. Speech and Audio Processing*, 8:37–50, 2000.

[2] Michael Collins. Parameter estimation for statistical parsing models: Theory and practice of distribution-free methods. In *IWPT*, 2001. available at http://www.ai.mit.edu/people/mcollins/publications.html.

[3] Koby Crammer and Yoram Singer. On the algorithmic implementation of multiclass kernel-based vector machines. *Journal of Machine Learning Research*, 2:265–292, 2001.

[4] Y. Lee, Y. Lin, and G. Wahba. Multicategory support vector machines, theory, and application to the classification of microarray data and satellite radiance data. *Journal of American Statistical Association*, 99:67–81, 2004.

[5] R. Tyrrell Rockafellar. *Convex analysis*. Princeton University Press, Princeton, NJ, 1970.

[6] Ben Taskar, Carlos Guestrin, and Daphne Koller. Max-margin markov networks. In Sebastian Thrun, Lawrence Saul, and Bernhard Schölkopf, editors, *Advances in Neural Information Processing Systems 16*. MIT Press, Cambridge, MA, 2004.

[7] J. Weston and C. Watkins. Multi-class support vector machines. Technical Report CSD-TR-98-04, Royal Holloway, 1998.

[8] Tong Zhang. Leave-one-out bounds for kernel methods. *Neural Computation*, 15:1397–1437, 2003.

[9] Tong Zhang. Statistical analysis of some multi-category large margin classification methods. *Journal of Machine Learning Research*, 5:1225–1251, 2004.

[10] Tong Zhang. Statistical behavior and consistency of classification methods based on convex risk minimization. *The Annals of Statitics*, 32:56–85, 2004. with discussion.
